# Robust Learning of Chaotic Attractors

**Rembrandt Bakker***
Chemical Reactor Engineering
Delft Univ. of Technology
*r.bakker@stm.tudelft.nl*

**Jaap C. Schouten**
Chemical Reactor Engineering
Eindhoven Univ. of Technology
*J.C.Schouten@tue.nl*

**Marc-Olivier Coppens**
Chemical Reactor Engineering
Delft Univ. of Technology
*coppens@stm.tudelft.nl*

**Floris Takens**
Dept. Mathematics
University of Groningen
*F.Takens@math.rug.nl*

**C. Lee Giles**
NEC Research Institute
Princeton NJ
*giles@research.nj.nec.com*

**Cor M. van den Bleek**
Chemical Reactor Engineering
Delft Univ. of Technology
*vdbleek@stm.tudelft.nl*

## Abstract

A fundamental problem with the modeling of chaotic time series data is that minimizing short-term prediction errors does *not guarantee* a match between the reconstructed attractors of model and experiments. We introduce a modeling paradigm that simultaneously learns to short-term predict and to locate the outlines of the attractor by a new way of nonlinear principal component analysis. Closed-loop predictions are constrained to stay within these outlines, to prevent divergence from the attractor. Learning is exceptionally fast: parameter estimation for the 1000 sample laser data from the 1991 Santa Fe time series competition took less than a minute on a 166 MHz Pentium PC.

## 1 Introduction

We focus on the following objective: given a set of experimental data and the assumption that it was produced by a deterministic chaotic system, find a set of model equations that will produce a time-series with identical chaotic characteristics, having the same chaotic attractor. The common approach consists of two steps: (1) identify a model that makes accurate short-term predictions; and (2) generate a long time-series with the model and compare the nonlinear-dynamic characteristics of this time-series with the original, measured time-series.

Principe *et al.* [1] found that in many cases the model can make good short-term predictions but does *not* learn the chaotic attractor. The method would be greatly improved if we could minimize directly the difference between the reconstructed attractors of the model-generated and measured data, instead of minimizing prediction errors. However, we cannot reconstruct the attractor without first having a prediction model. Until now research has focused on how to optimize both step 1 and step 2. For example, it is important to optimize the prediction horizon of the model [2] and to reduce complexity as much as possible. This way it was possible to learn the attractor of the benchmark laser time series data from the 1991 Santa Fe

*DelftChemTech, Chemical Reactor Engineering Lab, Julianalaan 136, 2628 BL, Delft, The Netherlands; http://www.cpt.stm.tudelft.nl/cpt/cre/research/bakker/.

time series competition. While training a neural network for this problem, we noticed [3] that the attractor of the model fluctuated from a good match to a complete mismatch from one iteration to another. We were able to circumvent this problem by selecting exactly that model that matches the attractor. However, after carrying out more simulations we found that what we neglected as an unfortunate phenomenon [3] is really a fundamental limitation of current approaches.

An important development is the work of Principe *et al.* [4] who use Kohonen Self Organizing Maps (SOMs) to create a discrete representation of the state space of the system. This creates a partitioning of the input space that becomes an infrastructure for local (linear) model construction. This partitioning enables to verify if the model input is near the original data (*i.e.*, detect if the model is not extrapolating) without keeping the training data set with the model. We propose a different partitioning of the input space that can be used to (i) learn the outlines of the chaotic attractor by means of a new way of nonlinear Principal Component Analysis (PCA), and (ii) *enforce* the model never to predict outside these outlines. The nonlinear PCA algorithm is inspired by the work of Kambhatla and Leen [5] on local PCA: they partition the input space and perform local PCA in each region. Unfortunately, this introduces discontinuities between neighboring regions. We resolve them by introducing a hierarchical partitioning algorithm that uses fuzzy boundaries between the regions. This partitioning closely resembles the hierarchical mixtures of experts of Jordan and Jacobs [6].

In Sec. 2 we put forward the fundamental problem that arises when trying to learn a chaotic attractor by creating a short-term prediction model. In Sec. 3 we describe the proposed partitioning algorithm. In Sec. 4 it is outlined how this partitioning can be used to learn the outline of the attractor by defining a potential that measures the distance to the attractor. In Sec. 5 we show modeling results on a toy example, the logistic map, and on a more serious problem, the laser data from the 1991 Santa Fe time series competition. Section 6 concludes.

## 2 The attractor learning dilemma

Imagine an experimental system with a chaotic attractor, and a time-series of noise-free measurements taken from this system. The data is used to fit the parameters of the model $\vec{z}_{t+1} = F_{\vec{w}}(\vec{z}_t, \vec{z}_{t-1}, \ldots, \vec{z}_{t-m})$ where F is a nonlinear function, $\vec{w}$ contains its adjustable parameters and $m$ is a positive constant. What happens if we fit the parameters $\vec{w}$ by nonlinear least squares regression? Will the model be stable, *i.e.*, will the closed-loop long term prediction converge to the same attractor as the one represented by the measurements?

Figure 1 shows the result of a test by Diks *et al.* [7] that compares the difference between the model and measured attractor. The figure shows that while the neural network is trained to predict chaotic data, the model quickly converges to the measured attractor (S=0), but once in a while, from one iteration to another, the match between the attractors is lost.

To understand what causes this instability, imagine that we try to fit the parameters of a model $\vec{z}_{t+1} = \vec{a} + B\vec{z}_t$ while the real system has a point attractor, $\vec{z} = \vec{\alpha}$, where $\vec{z}$ is the state of the system and $\vec{\alpha}$ its attracting value. Clearly, measurements taken from this system contain no information to

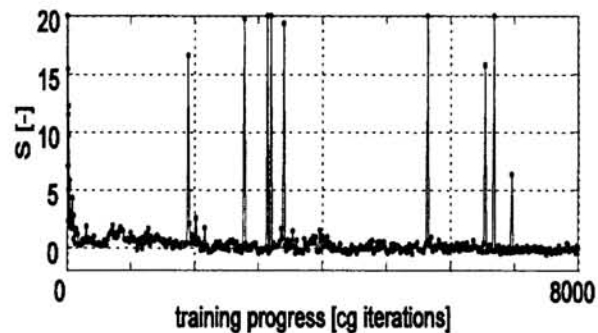

Figure 1: Diks test monitoring curve for a neural network model trained on data from an experimental chaotic pendulum [3].

estimate both $\vec{a}$ and $B$. If we fit the model parameters with non-robust linear least squares, $B$ may be assigned any value and if its largest eigenvalue happens to be greater than zero, the model will be unstable!

For the linear model this problem has been solved a long time ago with the introduction of singular value decomposition. There still is a need for a nonlinear counterpart of this technique, in particular since we have to work with very flexible models that are designed to fit a wide variety of nonlinear shapes, see for example the early work of Lapedes and Farber [8]. It is already common practice to control the complexity of nonlinear models by pruning or regularization. Unfortunately, these methods do not always solve the attractor learning problem, since there is a good chance that a nonlinear term explains a lot of variance in one part of the state space, while it causes instability of the attractor (without affecting the one-step-ahead prediction accuracy) elsewhere. In Secs. 3 and 4 we will introduce a new method for nonlinear principal component analysis that will detect and prevent unstable behavior.

## 3. The split and fit algorithm

The nonlinear regression procedure of this section will form the basis of the nonlinear principal component algorithm in Sec. 4. It consists of (i) a partitioning of the input space, (ii) a local linear model for each region, and (iii) fuzzy boundaries between regions to ensure global smoothness. The partitioning scheme is outlined in Procedure 1:

---
### Procedure 1: Partitioning the input space
---

1) Start with the entire set $Z$ of input data

2) Determine the direction of largest variance of $Z$: perform a singular value decomposition of $Z$ into the product $U\Sigma V^T$ and take the eigenvector (column of $V$) with the largest singular value (on the diagonal of $\Sigma$).

3) Split the data in two subsets (to be called: clusters) by creating a plane perpendicular to the direction of largest variance, through the center of gravity of $Z$.

4) Next, select the cluster with the largest sum squared error to be split next, and recursively apply 2-4 until a stopping criteria is met.

Figures 2 and 3 show examples of the partitioning. The disadvantage of dividing regression problems into localized subproblems was pointed out by Jordan and Jacobs [6]: the spread of the data in each region will be much smaller than the spread of the data as a whole, and this will increase the variance of the model parameters. Since we always split perpendicular to the direction of maximum variance, this problem is minimized.

The partitioning can be written as a binary tree, with each non-terminal node being a split and each terminal node a cluster. Procedure 2 creates fuzzy boundaries between the clusters.

---
### Procedure 2. Creating fuzzy boundaries
---

1) An input $\vec{z}$ enters the tree at the top of the partitioning tree.

2) The Euclidean distance to the splitting hyperplane is divided by the bandwidth $\beta$ of the split, and passed through a sigmoidal function with range [0,1]. This results in $\vec{z}$'s share $\sigma$ in the subset on $\vec{z}$'s side of the splitting plane. The share in the other subset is $1-\sigma$.

3) The previous step is carried out for all non-terminal nodes of the tree.

4) The membership $\mu_c$ of $\vec{z}$ to subset (terminal node) $c$ is computed by taking the product of all previously computed shares $\sigma$ along the path from the terminal node to the top of the tree.

If we would make all parameters adjustable, that is (i) the orientation of the splitting hyperplanes, (ii) the bandwidths $\beta$, and (iii) the local linear model parameters, the above model structure would be identical to the *hierarchical mixtures of experts* of Jordan and Jacobs [6]. However, we already fixed the hyperplanes and use Procedure 3 to compute the bandwidths:

### Procedure 3. Computing the Bandwidths

1) The bandwidths of the terminal nodes are taken to be a constant (we use 1.65, the 90% confidence limit of a normal distribution) times the variance of the subset *before it was last split*, in the direction of the eigenvector of that last split.

2) The other bandwidths do depend on the input $\vec{z}$. They are computed by climbing upward in the tree. The bandwidth of node $n$ is computed as a weighted sum between the $\beta$s of its right and left child, by the implicit formula $\beta_n = \sigma_L \beta_L \; \sigma_R \beta_R$, in which $\sigma_L$ and $\sigma_R$ depend on $\beta_n$. Starting from initial guess $\beta_n = \beta_L$ if $\sigma_L > 0.5$, or else $\beta_n = \beta_R$, the formula is solved in a few iterations.

This procedure is designed to create large overlap between neighboring regions and almost no overlap between non-neighboring regions. What remains to be fitted is the set of the local linear models. The $j$-th output of the split&fit model for a given input $\vec{z}_p$ is computed:

$$\hat{y}_{j,p} = \sum_{c=1}^{C} \mu_p^c \{\vec{a}_j^c \vec{z}_p + b_j^c\}, \text{where } \vec{a}^c \text{ and } b^c \text{ contain the linear model parameters of subset } c,$$

and $C$ is the number of clusters. We can determine the parameters of all local linear models in one global fit that is linear in the parameters. However, we prefer to locally optimize the parameters for two reasons: (i) it makes it possible to locally control the stability of the attractor and do the principal component analysis of Sec. 4; and (ii) the computing time for a linear regression problem with $r$ regressors scales $\sim O(r^3)$. If we would adopt global fitting, $r$ would scale linearly with $C$ and, while growing the model, the regression problem would quickly become intractable. We use the following iterative local fitting procedure instead.

### Procedure 4. Iterative Local Fitting

1) Initialize a $J$ by $N$ matrix of residuals $R$ to zero, $J$ being the number of outputs and $N$ the number of data.

2) For cluster $c$, if an estimate for its linear model parameters already exists, for each input vector $\vec{z}_p$ add $\mu_p^c \hat{y}_{j,p}$ to the matrix of residuals, otherwise add $\mu_p^c y_{j,p}$ to $R$, $y_{j,p}$ being the $j$-th element of the desired output vector for sample $p$.

3) Least squares fit the linear model parameters of cluster $c$ to predict the current residuals $R$, and subtract the (new) estimate, $\mu_p^c \hat{y}_{j,p}$, from $R$.

4) Do 2-4 for each cluster and repeat the fitting several times (default: 3).

From simulations we found that the above fast optimization method converges to the global minimum if it is repeated many times. Just as with neural network training, it is often better to use early stopping when the prediction error on an independent test set starts to increase.

## 4. Nonlinear Principal Component Analysis

To learn a chaotic attractor from a single experimental time-series we use the *method of delays*: the state $\bar{z}$ consists of $m$ delays taken from the time series. The embedding dimension $m$ must be chosen large enough to ensure that it contains sufficient information for faithful reconstruction of the chaotic attractor, see Takens [9]. Typically, this results in an $m$-dimensional state space with all the measurents covering only a much lower dimensional, but non-linearly shaped, subspace. This creates the danger pointed out in Sec. 2: the stability of the model in directions perpendicular to this low dimensional subspace cannot be guaranteed.

With the split & fit algorithm from Sec. 3 we can learn the non-linear shape of the low dimensional subspace, and, if the state of the system escapes from this subspace, we use the algorithm to redirect the state to the nearest point on the subspace. See Malthouse [10] for limitations of existing nonlinear PCA approaches. To obtain the low dimensional subspace, we proceed according to Procedure 5.

---

### Procedure 5. Learning the Low-dimensional Subspace

1) Augment the output of the model with the $m$-dimensional state $\bar{z}$: the model will learn to predict its own input.

2) In each cluster $c$, perform a singular value decomposition to create a set of $m$ principal directions, sorted in order of decreasing explained variance. The result of this decomposition is also used in step 3 of Procedure 4.

3) Allow the local linear model of each cluster to use no more than $m_{red}$ of these principal directions.

4) Define a potential $P$ to be the squared Euclidian distance between the state $\bar{z}$ and its prediction by the model.

---

The potential $P$ implicitly defines the lower dimensional subspace: if a state $\bar{z}$ is on the subspace, $P$ will be zero. $P$ will increase with the distance of $\bar{z}$ from the subspace. The model has learned to predict its own input with small error, meaning that it has tried to reduce $P$ as much as possible at exactly those points in state space where the training data was sampled. In other words, $P$ will be low if the input $\bar{z}$ is close to one of the original points in the training data set. From the split&fit algorithm we can analytically compute the gradient $dP/d\bar{z}$. Since the evaluation of the split&fit model involves a backward (computing the bandwidths) and forward pass (computing memberships), the gradient algorithm involves a forward and backward pass through the tree. The gradient is used to project states that are off the nonlinear subspace onto the subspace

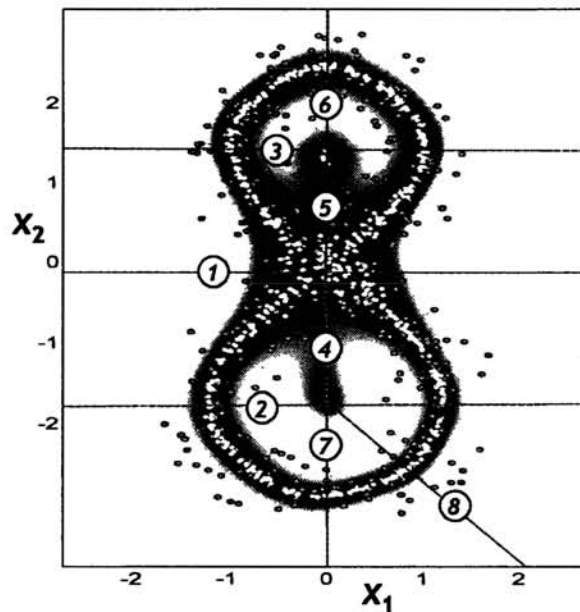

Figure 2. Projecting two-dimensional data on a one-dimensional self-intersecting subspace. The colorscale represents the potential $P$, white indicates $P>0.04$..

in one or a few Newton-Rhapson iterations. Figure 2 illustrates the algorithm for the problem of creating a one-dimensional representation of the number '8'. The training set consists of 136 clean samples, and Fig. 2 shows how a set of 272 noisy inputs is projected by a 48 subset split&fit model onto the one-dimensional subspace. Note that the center of the '8' cannot be well represented by a one-dimensional space. We leave development of an algorithm that automatically detects the optimum local subspace dimension for future research.

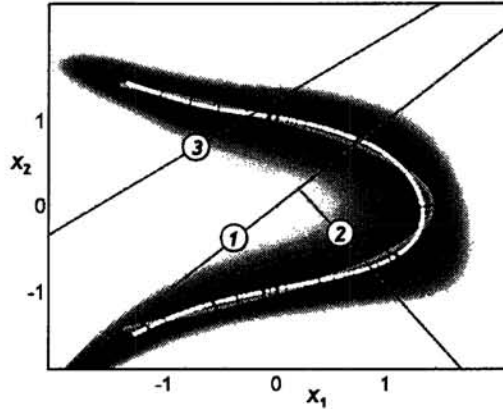

Figure 3. Learning the attractor of the two-input logistic map. The order of creation of the splits is indicated. The colorscale represents the potential $P$, white indicates $P > 0.05$.

## 5. Application Examples

First we show the nonlinear principal component analysis result for a toy example, the logistic map $z_{t+1} = 4z_t(1-z_t)$. If we use a model $z_{t+1} = F_{\bar{w}}(z_t)$, where the prediction only depends on one previous output, there is no lower dimensional space to which the attractor is confined. However, if we allow the output to depend on more than a single delay, we create a possibility for unstable behavior. Figure 3 shows how well the split&fit algorithm learns the one-dimensional shape of the attractor after creating only five regions. The parabola is slightly deformed (seen from the white lines perpendicular to the attractor), but this may be solved by increasing the number of splits.

Next we look at the laser data. The complex behavior of chaotic systems is caused by an interplay of destabilizing and stabilizing forces: the destabilizing forces make nearby points in state space diverge, while the stabilizing forces keep the state of the system bounded. This process, known as 'stretching and folding', results in the attractor of the system: the set of points that the state of the system will visit after all transients have died out. In the case of the laser data this behavior is clear cut: destabilizing forces make the signal grow exponentially until the increasing amplitude triggers a collapse that reinitiates the sequence. We have seen in neural network based models [3] and in this study that it is very hard for the models to cope with the sudden collapses. Without the nonlinear subspace correction of Sec. 4, most of the

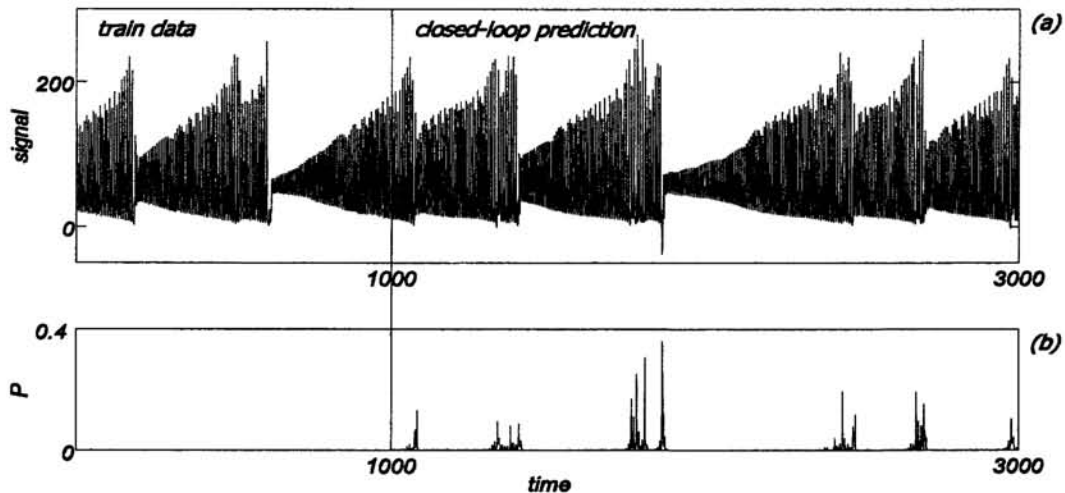

Figure 4. Laser data from the Santa Fe time series competition. The 1000 sample train data set is followed by iterated prediction of the model (a). After every prediction a correction is made to keep $P$ (see Sec. 4) small. Plot (b) shows $P$ before this correction.

models we tested grow without bounds after one or more rise and collapse sequences. That is not very surprising - the training data set contains only three examples of a collapse. Figure 4 shows how this is solved with the subspace correction: every time the model is about to grow to infinity, a high potential $P$ is detected (depicted in Fig. 3b) and the state of the system is directed to the nearest point on the subspace as learned from the nonlinear principal component analysis. After some trial and error, we selected an embedding dimension $m$ of 12 and a reduced dimension $m_{red}$ of 4. The split&fit model starts with a single dataset, and was grown until 48 subsets. At that point, the error on the 1000 sample train set was still decreasing rapidly but the error on an independent 1000 sample test set increased. We compared the reconstructed attractors of the model and measurements, using 9000 samples of closed-loop generated and 9000 samples of measured data. No significant difference between the two could be detected by the Diks test [7].

## 6. Conclusions

We present an algorithm that robustly models chaotic attractors. It simultaneously learns (1) to make accurate short term predictions; and (2) the outlines of the attractor. In closed-loop prediction mode, the state of the system is corrected after every prediction, to stay within these outlines. The algorithm is very fast, since the main computation is to least squares fit a set of local linear models. In our implementation the largest matrix to be stored is $N$ by $C$, $N$ being the number of data and $C$ the number of clusters. We see many applications other than attractor learning: the split&fit algorithm can be used as a fast learning alternative to neural networks and the new form of nonlinear PCA will be useful for data reduction and object recognition. We envisage to apply the technique to a wide range of applications, from the control and modeling of chaos in fluid dynamics to problems in finance and biology to fluid dynamics.

### Acknowledgements

This work is supported by the Netherlands Foundation for Chemical Research (SON) with financial aid from the Netherlands Organization for Scientific Research (NWO).

### References

[1] J.C. Principe, A. Rathie, and J.M. Kuo, "Prediction of Chaotic Time Series with Neural Networks and the Issue of Dynamic Modeling", *Int. J. Bifurcation and Chaos*, 2, 1992, p 989.

[2] J.M. Kuo, and J.C. Principe, "Reconstructed Dynamics and Chaotic Signal Modeling", In *Proc. IEEE Int'l Conf. Neural Networks*, 5, 1994, p 3131.

[3] R. Bakker, J.C. Schouten, C.L. Giles, F. Takens, C.M. van den Bleek, "Learning Chaotic Attractors by Neural Networks", submitted.

[4] J.C. Principe, L. Wang, M.A. Motter, "Local Dynamic Modeling with Self-Organizing Maps and Applications to Nonlinear System Identification and Control",*Proc. IEEE*, 86(11), 1998.

[5] N. Kambhatla, T.K. Leen, "Dimension Reduction by Local PCA", *Neural Computation*, 9, 1997, p. 1493

[6] M.I. Jordan, R.A. Jacobs, "Hierarchical Mixtures of Experts and the EM Algorithm", *Neural Compution*, 6, 1994, p. 181.

[7] C. Diks, W.R. van Zwet, F. Takens, and J. de Goede, "Detecting differences between delay vector distributions", *Physical Review E*, 53, 1996, p. 2169.

[8] A. Lapedes, R. Farber, "Nonlinear Signal Processing Using Neural Networks: Prediction and System Modelling", *Los Alamos Technical Report* LA-UR-87-2662.

[9] F. Takens, "Detecting strange attractors in turbulence", *Lecture notes in Mathematics*, 898, 1981, p. 365.

[10] E.C. Malthouse, "Limitations of Nonlinear PCA as performed with Generic Neural Networks, *IEEE Trans. Neural Networks*, 9(1), 1998, p. 165.